# Algebraic Set Kernels with Application to Inference Over Local Image Representations

**Amnon Shashua**   and   **Tamir Hazan** *

## Abstract

This paper presents a general family of algebraic positive definite similarity functions over spaces of matrices with varying column rank. The columns can represent local regions in an image (whereby images have varying number of local parts), images of an image sequence, motion trajectories in a multibody motion, and so forth. The family of set kernels we derive is based on a group invariant tensor product lifting with parameters that can be naturally tuned to provide a cook-book of sorts covering the possible "wish lists" from similarity measures over sets of varying cardinality. We highlight the strengths of our approach by demonstrating the set kernels for visual recognition of pedestrians using local parts representations.

## 1   Introduction

In the area of learning from observations there are two main paths that are often mutually exclusive: (i) the design of learning algorithms, and (ii) the design of data representations. The algorithm designers take pride in the fact that their algorithm can generalize well given straightforward data representations (most notable example is SVM [11]), whereas those who work on data representations demonstrate often remarkable results with sophisticated data representations using only straightforward learning algorithms (e.g. [5, 10, 6]). This dichotomy is probably most emphasized in the area of computer vision, where image understanding from observations involve data instances of images or image sequences containing huge amounts of data. A straightforward representation treating all the measurements as a single vector, such as the raw pixel data, or a transformed raw-pixel data, places unreasonable demands on the learning algorithm. The "holistic" representations suffer also from sensitivity to occlusions, invariance to local and global transformations, non-rigidity of local parts of the object, and so forth.

Practitioners in the area of data representations have long noticed that a collection of local representations (part-based representations) can be most effective to ameliorate changes of appearance [5, 10, 6]. The local data representations vary in their sophistication, but share the same principle where an image corresponds to a collection of points each in a relatively small dimensional space — instead of a single point in high-dimensional space induced by holistic representations. In general, the number of points (local parts) per image may vary and the dimension of each point may vary as well. The local representations tend

to be robust against occlusions, local and global transformations and preserve the original resolution of the image (the higher the resolution the more parts are generated per image).

The key for unifying local and holistic representations for inference engines is to design positive definite similarity functions (a.k.a. kernels) over *sets* (of vectors) of varying cardinalities. A Support Vector Machine (SVM) [11] can then handle sets of vectors as a single instance via application of those "set kernels". A set kernel would be useful also to other types of inference engines such as kernel versions of PCA, LDA, CCA, ridge regression and any algorithm which can be mapped onto inner-products between pairs of data instances (see [8] for details on kernel methods).

Formally, we consider an instance being represented by a collection of vectors, which for the sake of convenience, form the columns of a matrix. We would like to find an algebraic family of similarity functions $sim(A, B)$ over matrices $A, B$ which satisfy the following requirements: (i) $sim(A, B)$ is an inner product, i.e., $sim(A, B) = \phi(A)^\top \phi(B)$ for some mapping $\phi()$ from matrices to vectors, (ii) $sim(A, B)$ is built over local kernel functions $k(\mathbf{a}_i, \mathbf{b}_j)$ over columns $\mathbf{a}_i$ and $\mathbf{b}_j$ of $A, B$ respectively, (iii) The column cardinality (rank of column space) of $A$ and $B$ need not be the same (number of local parts may differ from image to image), and (iv) the parameters of $sim(A, B)$ should induce the properties of invariance to order (alignement) of parts, part occlusions, and degree of interactions between local parts. In a nutshell, our work provides a cook-book of sorts which *fundamentally covers the possible algebraic kernels over collections of local representations built on top of local kernels* by combining (linearly and non-linearly) local kernels to form a family of global kernels over local representations.

The design of a kernel over sets of vectors has been recently attracting much attention in the computer vision and machine learning literature. A possible approach is to fit a distribution to the set of vectors and define the kernel as a distribution matching measure [9, 12, 4]. This has the advantage that the number of local parts can vary but at the expense of fitting a distribution to the variation over parts. The variation could be quite complex at times, unlikely to fit into a known family of distributions in many situations of interest, and in practice the sample size (number of columns of $A$) is not sufficiently large to reliably fit a distribution. The alternative, which is the approach taken in this paper, is to create a kernel over sets of vectors in a direct manner. When the column cardinality is equal it is possible to model the similarity measure as a function over the principal angles between the two column spaces ([14] and references therein) while for varying column cardinality only heuristic similarity measures (which are not positive definite) have so far been introduced [13].

It is important to note that although we chose SVM over local representations as the application to demonstrate the use of set kernels, the need for adequately working with instances made out of sets of various cardinalities spans many other application domains. For example, an image sequence may be represented by a set (ordered or unordered) of vectors, where each vector stands for an image, the pixels in an image can be represented as a tuple consisting of position, intensity and other attributes, motion trajectories of multiply moving bodies can be represented as a collection of vectors, and so on. Therefore, the problem addressed in this paper is fundamental both theoretically and from a practical perspective as well.

## 2 The General Family of Inner-Products over Matrices

We wish to derive the general family of positive definite similarity measures $sim(A, B)$ over matrices $A, B$ which have the same number of rows but possibly different column rank (in particular, different number of columns). Let $A$ be of dimensions $n \times k$ and $B$ of dimension $n \times q$ where $n$ is fixed and $k, q$ can vary at will over the application of $sim(\cdot, \cdot)$ on pairs of matrices. Let $m = \max\{n, k, q\}$ be the upper bound over all values

of $k, q$ encountered by the data. Let $\mathbf{a}_i, \mathbf{b}_j$ be the column vectors of matrices $A, B$ and let $k(\mathbf{a}_i, \mathbf{b}_j)$ be the local kernel function. For example, in the context where the column vectors represent local parts of an image, then the matching function $k(\cdot, \cdot)$ between pairs of local parts provides the building blocks of the overall similarity function. The local kernel is some positive definite function $k(\mathbf{x}, \mathbf{y}) = \phi(\mathbf{x})^\top \phi(\mathbf{y})$ which is the inner-product between the "feature"-mapped vectors $\mathbf{x}, \mathbf{y}$ for some feature map $\phi(\cdot)$. For example, if $\phi(\cdot)$ is the polynomial map of degree up to $d$, then $k(\mathbf{x}, \mathbf{y}) = (1 + \mathbf{x}^\top \mathbf{y})^d$.

The local kernels can be combined in a linear or non-linear manner. When the combination is linear the similarity becomes the analogue of the inner-product between vectors extended to matrices. We will refer to the linear family as $sim(A, B) = < A, B >$ and that will be the focus of this section. In the next section we will derive the general (algebraic) non-linear family which is based on "lifting" the input matrices $A, B$ onto higher dimensional spaces and feeding the result onto the $< \cdot, \cdot >$ machinery developed in this section, i.e., $sim(A, B) = < \psi(A), \psi(B) >$.

We will start by embedding $A, B$ onto $m \times m$ matrices by zero padding as follows. Let $\mathbf{e}_i$ denote the i'th standard basis vector $(0, .., 0, 1, 0, .., 0)$ of $R^m$. The the embedding is represented by linear combinations of tensor products:

$$ A \to \sum_{i=1}^{n} \sum_{j=1}^{k} a_{ij} \mathbf{e}_i \otimes \mathbf{e}_j, \qquad B \to \sum_{l=1}^{n} \sum_{t=1}^{q} b_{lt} \mathbf{e}_l \otimes \mathbf{e}_t. $$

Note that $A, B$ are the upper-left blocks of the zero-padded matrices. Let $S$ be a positive semi definite $m^2 \times m^2$ matrix represented by $S = \sum_{r=1}^{P} G_r \otimes F_r$ where $G_r, F_r$ are $m \times m$ matrices[1]. Let $\hat{F}_r$ be the $q \times k$ upper-left sub-matrix of $F_r^\top$, and let $\hat{G}_r$ be the $n \times n$ upper-left sub-matrix of $G_r$. We will be using the following three identities:

$$ G\mathbf{x}_1 \otimes F\mathbf{x}_2 = (G \otimes F)(\mathbf{x}_1 \otimes \mathbf{x}_2), $$
$$ (G \otimes F)(G' \otimes F') = GG' \otimes FF', $$
$$ < \mathbf{x}_1 \otimes \mathbf{x}_2, \mathbf{y}_1 \otimes \mathbf{y}_2 > = (\mathbf{x}_1^\top \mathbf{y}_1)(\mathbf{x}_2^\top \mathbf{y}_2). $$

The inner-product $< A, B >$ over all p.s.d. matrices $S$ has the form:

$$
\begin{aligned}
< A, B > &= < \sum_{i,j} a_{ij} \mathbf{e}_i \otimes \mathbf{e}_j, (\sum_r G_r \otimes F_r) \sum_{l,t} b_{lt} \mathbf{e}_l \otimes \mathbf{e}_t > \\
&= \sum_r \sum_{i,j,l,t} a_{ij} b_{lt} < \mathbf{e}_i \otimes \mathbf{e}_j, G_r \mathbf{e}_l \otimes F_r \mathbf{e}_t > \\
&= \sum_r \sum_{i,j,l,t} a_{ij} b_{lt} (\mathbf{e}_i^\top G_r \mathbf{e}_l)(\mathbf{e}_j^\top F_r \mathbf{e}_t) \\
&= \sum_r \sum_{i,j,l,t} a_{ij} b_{lt} (G_r)_{il} (F_r)_{jt} \\
&= \sum_r \sum_{lt} (A^\top \hat{G}_r B)_{jt} (F_r)_{jt} \\
&= trace \left( \sum_r (A^\top \hat{G}_r B) \hat{F}_r \right)
\end{aligned}
$$

We have represented the inner product $< A, B >$ using the choice of $m \times m$ matrices $G_r, F_r$ instead of the choice of a single $m^2 \times m^2$ p.s.d. matrix $S$. The matrices $G_r, F_r$

must be selected such that $\sum_{r=1}^{p} G_r \otimes F_r$ is positive semi definite. The problem of deciding on the the necessary conditions on $F_r$ and $G_r$ such that the sum over tensor products is p.s.d is difficult. Even deciding whether a given $S$ has a separable decomposition is known to be NP-hard [3]. The sufficient conditions are easy — choosing $G_r, F_r$ to be positive semi definite would make $\sum_{r=1}^{p} G_r \otimes F_r$ positive semi definite as well. In this context (of separable $S$) we need one more constraint in order to work with non-linear local kernels $k(\mathbf{x}, \mathbf{y}) = \phi(\mathbf{x})^\top \phi(\mathbf{y})$: the matrices $\hat{G}_r = \tilde{M}_r^\top \tilde{M}_r$ must "distribute with the kernel", namely there exist $M_r$ such that

$$k(M_r\mathbf{x}, M_r\mathbf{y}) = \phi(M_r\mathbf{x})^\top \phi(M_r\mathbf{y}) = \phi(\mathbf{x})^\top \tilde{M}_r^\top \tilde{M}_r \phi(\mathbf{y}) = \phi(\mathbf{x})^\top \hat{G}_r \phi(\mathbf{y}).$$

To summarize the results so far, the most general, but seperable, analogue of the inner-product over vectors to the inner-product of matrices of varying column cardinality has the form:

$$< A, B >= \sum_r trace(H_r \hat{F}_r) \tag{1}$$

Where the entries of $H_r$ consists of $k(M_r\mathbf{a}_i, M_r\mathbf{b}_j)$ over the columns of $A, B$ after possibly undergoing global coordinate changes by $M_r$ (the role of $\hat{G}_r$), and $\hat{F}_r$ are the $q \times k$ upper-left sub-matrix of positive definite $m \times m$ matrices $F_r^\top$.

The role of the matrices $\hat{G}_r$ is to perform global coordinate changes of $R^n$ before application of the kernel $k()$ on the columns of $A, B$. These global transformations include projections (say onto prototypical "parts") that may be given or "learned" from a training set. The matrices $\hat{F}_r$ determine the range of interaction between columns of $A$ and columns of $B$. For example, when $\hat{G}_r = I$ then $< A, B >= trace(A^\top B\hat{F})$ where $\hat{F}$ is the upper-left submatrix with the appropriate dimension of some fixed $m \times m$ p.s.d matrix $F = \sum_r F_r$. Note that entries of $A^\top B$ are $k(\mathbf{a}_i, \mathbf{b}_j)$. In other words, when $G_r = I$, $< A, B >$ boils down to a simple linear super-position of the local kernels, $\sum_{ij} k(\mathbf{a}_i, \mathbf{b}_j) f_{ij}$ where the entries $f_{ij}$ are part of the upper-left block of a fixed positive definite matrix $F$ where the block dimensions are commensurate with the number of columns of $A$ and those of $B$. The various choices of $F$ determine the type of *invariances* one could obtain from the similarity measure. For example, when $F = I$ the similarity is simply the sum (average) of the local kernels $k(\mathbf{a}_i, \mathbf{b}_i)$ thereby assuming we have a strict *alignment* between the local parts represented by $A$ and the local parts represented by $B$. On the other end of the invariance spectrum, when $F = \mathbf{1}\mathbf{1}^\top$ (all entries are "1") the similarity measure averages over all interactions of local parts $k(\mathbf{a}_i, \mathbf{b}_j)$ thereby achieving an *invariance* to the order of the parts. A *decaying* weighted interaction such as $f_{ij} = \sigma^{-|i-j|}$ would provide a middle ground between the assumption of strict alignment and the assumption of complete lack of alignment. In the section below we will derive the non-linear version of $sim(A, B)$ based on the basic machinery of $< A, B >$ of eqn. (1) and lifting operations on $A, B$.

## 3   Lifting Matrices onto Higher Dimensions

The family of $sim(A, B) =< A, B >$ forms a weighted linear superposition of the local kernel $k(\mathbf{a}_i, \mathbf{b}_j)$. Non-linear combinations of local kernels emerge using mappings $\psi(A)$ from the input matrices onto other higher-dimensional matrices, thus forming $sim(A, B) =< \psi(A), \psi(B) >$. Additional invariance properties and parameters controlling the perfromance of $sim(A, B)$ emerge with the introduction of non-linear combinations of local kernels, and those will be discussed later on in this section.

Consider the general d-fold lifting $\psi(A) = A^{\otimes d}$ which can be viewed as a $n^d \times k^d$ matrix. Let $F_r$ be a p.s.d. matrix of dimension $m^d \times m^d$ and $\hat{F}_r$ be the upper-left $q^d \times k^d$ block of $F_r$. Let $G_r = (\hat{G}_r)^{\otimes d}$ be a p.s.d matrix of dimension $n^d \times n^d$ where $\hat{G}_r$ is p.s.d. $n \times n$ matrix. Using the identity $(A^{\otimes d})^\top B^{\otimes d} = (A^\top B)^{\otimes d}$ we obtain the inner-product in the

lifted space:

$$< A^{\otimes d}, B^{\otimes d} > = \sum_r trace\left((A^\top \hat{G}_r B)^{\otimes d} \hat{F}_r\right).$$

By taking linear combinations of $< A^{\otimes l}, B^{\otimes l} >$, $l = 1, ..., d$, we get the general non-homogenous d-fold inner-product $sim^d(A, B)$. A this point the formulation is general but somewhat unwieldy computational-wise. The key for computational simplification lay in the fact that choices of $F_r$ determine not only local interactions (as in the linear case) but also *group invariances*. The group invariances are a result of *applying symmetric operators on the tensor product space* — we will consider two of those operators here, known as the the d-fold alternating tensor $A^{\wedge d} = A \wedge .... \wedge A$ and the d-fold symmetric tensor $A^{\cdot d} = A \cdot ... \cdot A$. These lifting operations introduce the *determinant* and *permanent* operations on submatrices of $A^\top \hat{G}_r B$, as described below.

The alternating tensor is a multilinear map of $R^n$, $(A \wedge .... \wedge A)(\mathbf{x}_1 \wedge ... \wedge \mathbf{x}_d) = A\mathbf{x}_1 \wedge ... \wedge A\mathbf{x}_d$, where

$$\mathbf{x}_1 \wedge ... \wedge \mathbf{x}_d = \frac{1}{d!} \sum_{\sigma \in S_d} sign(\sigma)\mathbf{x}_{\sigma(1)} \otimes .... \otimes \mathbf{x}_{\sigma(d)},$$

where $S_d$ is the symmetric group over $d$ letters and $\sigma \in S_d$ are the permutations of the group. If $\mathbf{x}_1, ..., \mathbf{x}_n$ form a basis of $R^n$, then the $\binom{n}{d}$ elements $\mathbf{x}_{i_1} \wedge ... \wedge \mathbf{x}_{i_d}$, where $1 \leq i_1 < ... < i_d \leq n$ form a basis of the alternating $d - fold$ tensor product of $R^n$, denoted as $\Lambda^d R^n$. If $A \in R^{n \times k}$ is a linear map on $R^n$ sending points to $R^k$, then $A^{\wedge d}$ is a linear map on $\Lambda^d R^n$ sending $\mathbf{x}_1 \wedge ... \wedge \mathbf{x}_d$ to $A\mathbf{x}_1 \wedge ... \wedge A\mathbf{x}_d$, i.e., sending points in $\Lambda^d R^n$ to points in $\Lambda^d R^k$. The matrix representation of $A^{\wedge d}$ is called the "d'th compound matrix" $C_d(A)$ whose $(i_1, ..., i_d | j_1, ..., j_d)$ entry has the value $det(A[i_1, ..., i_d : j_1, ..., j_d])$ where the determinant is of the $d \times d$ block constructed by choosing the rows $i_1, ..., i_d$ and the columns $j_1, ..., j_d$ of $A$. In other words, $C_d(A)$ has $\binom{n}{d}$ rows and $\binom{k}{d}$ columns (instead of $n^d \times k^d$ necessary for $A^{\otimes d}$) whose entries are equal to the $d \times d$ minors of $A$. When $k = d$, $C_k(A)$ is a vector known as the Grasmanian of $A$, and when $n = k = d$ then $C_d(A) = det(A)$. Finally, the identity $(A^{\otimes d})^\top B^{\otimes d} = (A^\top B)^{\otimes d}$ specializes to $(A^{\wedge d})^\top B^{\wedge d} = (A^\top B)^{\wedge d}$ which translates to the identity $C_d(A)^\top C_d(B) = C_d(A^\top B)$ known as the Binet-Cauchy theorem [1]. Taken together, the "d-fold alternating kernel" $\Lambda^d(A, B)$ is defined by:

$$\Lambda^d(A, B) = < A^{\wedge d}, B^{\wedge d} > = < C_d(A), C_d(B) > = \sum_r trace\left(C_d(A^\top \hat{G}_r B)\hat{F}_r\right), \quad (2)$$

where $\hat{F}_r$ is the $\binom{q}{d} \times \binom{k}{d}$ upper-left submatrix of the p.s.d $\binom{m}{d} \times \binom{m}{d}$ matrix $F_r$. Note that the local kernel plugs in as the entries of $(A^\top \hat{G}_r B)_{ij} = k(M_r \mathbf{a}_i, M_r \mathbf{b}_j)$ where $\hat{G}_r = M_r^\top M_r$.

Another symmetric operator on the tensor product space is via the d-fold symmetric tensor space $Sym^d R^n$ whose points are:

$$\mathbf{x}_1 \cdots \mathbf{x}_d = \frac{1}{d!} \sum_{\sigma \in S_d} \mathbf{x}_{\sigma(1)} \otimes .... \otimes \mathbf{x}_{\sigma(d)}.$$

The analogue of $C_d(A)$ is the "d'th power matrix" $R_d(A)$ whose $(i_1, ..., i_d | j_1, ..., j_d)$ entry has the value $perm(A[i_1, ..., i_d : j_1, ..., j_d])$ and which stands for the map $A^{\cdot d}$

$$(A \cdots A)(\mathbf{x}_1 \cdots \mathbf{x}_d) = A\mathbf{x}_1 \cdots A\mathbf{x}_d.$$

In other words, $R_d(A)$ has $\binom{n+d-1}{d}$ rows and $\binom{k+d-1}{d}$ columns whose entries are equal to the $d \times d$ *permanents* of $A$. The analogue of the Binet-Cauchy theorem is $R_d(A)^\top R_d(B) =$

$R_d(A^\top B)$. The ensuing kernel similarity function, referred to as the "d-fold symmetric kernel" is:

$$Sym^d(A, B) = < A^{\cdot d}, B^{\cdot d} > = < R_d(A), R_d(B) > = \sum_r trace\left(R_d(A^\top \hat{G}_r B)\hat{F}_r\right) \quad (3)$$

where $\hat{F}_r$ is the $\binom{q+d-1}{d} \times \binom{k+d-1}{d}$ upper-left submatrix of the positive definite $\binom{m+d-1}{d} \times \binom{n+d-1}{d}$ matrix $F_r$. Due to lack of space we will stop here and spend the remainder of this section in describing in laymen terms what are the properties of these similarity measures, how they can be constructed in practice and in a computationally efficient manner (despite the combinatorial element in their definition).

### 3.1 Practical Considerations

To recap, the family of similarity functions $sim(A, B)$ comprise of the linear version $< A, B >$ (eqn. 1) and non-linear versions $\Lambda^l(A, B), Sym^l(A, B)$ (eqns. 2,3) which are group projections of the general kernel $< A^{\otimes d}, B^{\otimes d} >$. These different similarity functions are controlled by the choice of three items: $G_r, F_r$ and the parameter $d$ representing the degree of the tensor product operator. Specifically, we will focus on the case $G_r = I$ and on $\Lambda^d(A, B)$ as a representative of the non-linear family. The role of $\hat{G}_r$ is fairly interesting as it can be viewed as a projection operator from "parts" to prototypical parts that can be learned from a training set but we leave this to the full length article that will appear later.

Practically, to compute $\Lambda^d(A, B)$ one needs to run over all $d \times d$ blocks of the $k \times q$ matrix $A^\top B$ (whose entries are $k(\mathbf{a}_i, \mathbf{b}_j)$) and for each block compute the determinant. The similarity function is a weighted sum of all those determinants weighted by $f_{ij}$. By appropriate selection of $F$ one can control both the complexity (avoid running over all possible $d \times d$ blocks) of the computation and the degree of interaction between the determinants. These determinants have an interesting geometric interpretation if those are computed over unitary matrices — as described next.

Let $A = Q_A R_A$ and $B = Q_B R_B$ be the QR factorization of the matrices, i.e., $Q_A$ has orthonormal columns which span the column space of $A$, then it has been recently shown [14] that $R_A^{-1}$ can be computed from $A$ using only operations over $k(\mathbf{a}_i, \mathbf{a}_j)$. Therefore, the product $Q_A^\top Q_B$, which is equal to $R_A^{-T} A^\top B R_B^{-1}$, can be computed using only local kernel applications. In other words, for each $A$ compute $R_A^{-1}$ (can be done using only inner-products over columns of $A$), then when it comes to compute $A^\top B$ compute instead $R_A^{-T} A^\top B R_B^{-1}$ which is equivalent to computing $Q_A^\top Q_B$. Thus effectively we have replaced every $A$ with $Q_A$ (unitary matrix).

Now, $\Lambda^d(Q_A, Q_B)$ for unitary matrices is the sum over the product of the cosine principal angles between $d$-dim subspaces spanned by columns of $A$ and $B$. The value of each determinant of the $d \times d$ blocks of $Q_A^\top Q_B$ is equal to the product of the cosine principal angles between the respective $d$-dim subspaces determined by corresponding selection of $d$ columns from $A$ and $d$ columns from $B$. For example, the case $k = q = d$ produces $\Lambda^d(Q_A, Q_B) = det(Q_A^\top Q_B)$ which is the product of the eigenvalues of the matrix $Q_A^\top Q_B$. Those eigenvalues are the cosine of the principal angles between the column space of $A$ and the column space of $B$ [2]. Therefore, $det(Q_A^\top Q_B)$ measures the "angle" between the two subspaces spanned by the respective columns of the input matrices — in particular is invariant to the order of the columns. For smaller values of $d$ we obtain the *sum over such products* between subspaces spanned by subsets of $d$ columns between $A$ and $B$.

The advantage of smaller values of $d$ is two fold: first it enables to compute the similarity when $k \neq q$ and second breaks down the similarity between subspaces into smaller pieces. The entries of the matrix $F$ determine which subspaces are being considered and the interaction between subspaces in $A$ and $B$. A diagonal $F$ compares corresponding subspaces

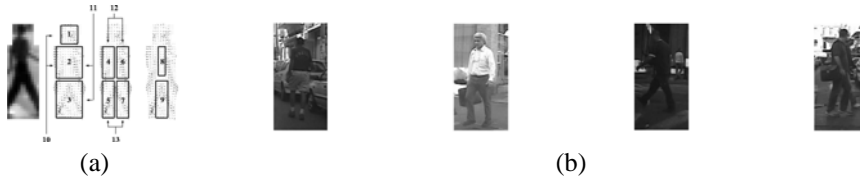

(a)                                          (b)

Figure 1: *(a) The configuration of the nine sub-regions is displayed over the gradient image. (b) some of the positive examples — note the large variation in appearance, pose and articulation.*

between $A$ and $B$ whereas off-diagonal entries would enable comparisons between different choices of subspaces in $A$ and in $B$. For example, we may want to consider choices of $d$ columns arranged in a "sliding" fashion, i.e., column sets $\{1, .., d\}, \{2, ..., d+1\}, ...$ and so forth, *instead of the combinatorial number of all possible choices*. This selection is associated with a sparse diagonal $F$ where the non-vanishing entries along the diagonal have the value of "1" and correspond to the sliding window selections.

To conclude, in the linear version $< A, B >$ the role of $F$ is to determine the range of interaction between columns of $A$ and columns of $B$, whereas with the non-linear version it is the interaction between $d$-dim *subspaces* rather than individual columns. We could select all possible interactions (exponential number) or any reduced interaction set such as the sliding window rule (linear number of choices) as described above.

## 4   Experiments

We examined the performance of $sim(A, B)$ on part-based representations for pedestrian detection using SVM for the inference engine. The dataset we used (courtesy of Mobileye Ltd.) covers a challenging variability of appearance, viewing position and body articulation (see Fig. 1). We ran a suit of comparative experiments using $sim(A, B) =< A, B >$ with three versions of $F = \{I, 11^{\top}, decay\}$ with local kernels covering linear, $d$'th degree polynomial ($d = 2, 6$) and RBF kernel, and likewise with $sim(A, B) = \Lambda^d(A, B)$ with $d = 2$, sparse diagonal $F$ (covering a sliding window configuration) and with linear, polynomial and RBF local kernels. We compared our results to the conventional down-sampled holistic representation where the raw images were down-sampled to size $20 \times 20$ and $32 \times 32$. Our tests also included simulation of occlusions (in the test images) in order to examine the sensitivity of our $sim(A, B)$ family to occlusions. For the local part representation, the input image was divided into 9 fixed regions where for each image local orientation statistics were were generated following [5, 7] with a total of 22 numbers per region (see Fig 1a), thereby making a $22 \times 9$ matrix representation to be fed into $sim(A, B)$. The size of the training set was 4000 split evenly between positive and negative examples and a test set of 4000 examples was used to evaluate the performance of each trial. The table below summarizes the accuracy results for the raw-pixel (holistic) representation over three trials: (i) images down-sampled to $20 \times 20$, (ii) images down-sampled to $32 \times 32$, and (iii) test images were partially occluded ($32 \times 32$ version). The accuracy figures are the ratio between the sum of the true positives and true negatives and the total number of test examples.

|  | raw linear | poly $d = 2$ | poly $d = 6$ | RBF |
|---|---|---|---|---|
| $20 \times 20$ | 78% | 83% | 84% | 86% |
| $32 \times 32$ | 78% | 84% | 85% | 82% |
| occlusion | 73.5% | 72% | 77% | 76.5% |

The table below displays $sim(A, B)$ with linear and RBF local kernels.

| local kernel | $< A, B >, F = I$ | $< A, B >, F = 11^{\top}$ | $< A, B >, f_{ij} = 2^{-\|i-j\|}$ | $\Lambda^2(A, B)$ |
|---|---|---|---|---|
| linear | 90.8% | 85% | 90.6% | 88% |
| RBF | 91.2% | 85% | 90.4% | 90% |

One can see that the local part representation provides a sharp increase in accuracy compared to the raw pixel holistic representation. The added power of invariance to order of parts induced by $< A, B >, F = 11^\top$ is not required since the parts are aligned and therefore the accuracy is the highest for the linear combination of local RBF $< A, B >, F = I$. The same applies for the non-linear version $\Lambda^d(A, B)$ — the additional invariances that come with a non-linear combination of local parts are apparently not required. The power of non-linearity associated with the combination of local parts comes to bear when the test images have occluded parts, i.e., at random one of the columns of the input matrix is removed (or replaced with a random vector), as shown in the table below:

| local kernel | $< A, B >, F = I$ | $\Lambda^2(A, B)$ |
|---|---|---|
| linear | 62% | 87% |
| RBF | 83% | 88% |

One can notice that a linear combination of local parts suffers from reduced accuracy whereas the non-linear combination maintains a stable accuracy (compare the right-most columns of the two tables above). Although the experiments above are still preliminary they show the power and potential of the $sim(A, B)$ family of kernels defined over local kernels. With the principles laid down in Section 3 one can construct a large number (we touched only a few) of algebraic kernels which combine the local kernels in non-linear ways thus creating invariances to order and increased performance against occlusion. Further research is required for sifting through the various possibilities with this new family of kernels and extracting their properties, their invariances and behavior under changing parameters $(F_r, G_r, d)$.

## Footnotes

*School of Engineering and Computer Science, Hebrew University of Jerusalem, Jerusalem 91904, Israel

[1] Any $S$ can be represented as a sum over tensor products: given column-wise ordering, the matrix $G \otimes F$ is composed of $n \times n$ blocks of the form $f_{ij} G$. Therefore, take $G_r$ to be the $n \times n$ blocks of $S$ and $F_r$ to be the elemental matrices which have "1" in coordinate $r = (i, j)$ and zero everywhere else.

## References

[1] A.C. Aitken. *Determinants and Matrices*. Interscience Publishers Inc., 4th edition, 1946.

[2] G.H. Golub and C.F. Van Loan. *Matrix computations*. John Hopkins University Press, 1989.

[3] L. Gurvits. Classical deterministic complexity of edmonds' problem and quantum entanglement. In *ACM Symp. on Theory of Computing*, 2003.

[4] R. Kondor and T. Jebara. A kernel between sets of vectors. In *International Conference on Machine Learning, ICML*, 2003.

[5] D.G. Lowe. Distinctive image features from scale-invariant keypoints. *International Journal of Computer Vision*, 2004.

[6] C. Schmidt and R. Mohr. Local grey-value invariants for image retrieval. *IEEE Transactions on Pattern Analysis and Machine Intelligence*, 19(5):530–535, 1997.

[7] A. Shashua, Y. Gdalyahu, G. Hayun and L. Mann. "Pedestrian Detection for Driving Assistance Systems". *IEEE Intelligent Vehicles Symposium (IV2004)*, June. 2004, Parma Italy.

[8] B. Schölkopf and A.J. Smola. *Learning with Kernels*. MIT Press, Cambridge, MA, 2002.

[9] G. Shakhnarovich, J.W. Fisher, and T. Darrell. Face recognition from long-term observations. In *Proceedings of the European Conference on Computer Vision*, 2002.

[10] S. Ullman, M. Vidal-Naquet, and E. Sali. Visual features of intermediate complexity and their use in classification. *Nature Neuroscience*, 5(7):1–6, 2002.

[11] V.N. Vapnik. *The nature of statistical learning*. Springer, 2nd edition, 1998.

[12] N. Vasconcelos, P. Ho, and P. Moreno. The kullback-leibler kernel as a framework for discriminant and localized representations for visual recognition. In *Proceedings of the European Conference on Computer Vision*, pages 430–441, Prague, Czech Republic, May 2004.

[13] C. Wallraven, B. Caputo, and A. Graf. Recognition with local features: the kernel recipe. In *Proceedings of the International Conference on Computer Vision*, 2003.

[14] L. Wolf and A. Shashua. Learning over sets using kernel principal angles. *Journal of Machine Learning Research (JMLR)*, 4(10):913–931, 2003.
